# Newton-Like Methods for Sparse Inverse Covariance Estimation

**Peder A. Olsen**
IBM, T. J. Watson Research Center
pederao@us.ibm.com

**Figen Oztoprak**
Sabanci University
figen@sabanciuniv.edu

**Jorge Nocedal**
Northwestern University
nocedal@eecs.northwestern.edu

**Steven J. Rennie**
IBM, T. J. Watson Research Center
sjrennie@us.ibm.com

## Abstract

We propose two classes of second-order optimization methods for solving the sparse inverse covariance estimation problem. The first approach, which we call the Newton-LASSO method, minimizes a piecewise quadratic model of the objective function at every iteration to generate a step. We employ the fast iterative shrinkage thresholding algorithm (FISTA) to solve this subproblem. The second approach, which we call the Orthant-Based Newton method, is a two-phase algorithm that first identifies an orthant face and then minimizes a smooth quadratic approximation of the objective function using the conjugate gradient method. These methods exploit the structure of the Hessian to efficiently compute the search direction and to avoid explicitly storing the Hessian. We also propose a limited memory BFGS variant of the orthant-based Newton method. Numerical results, including comparisons with the method implemented in the QUIC software [1], suggest that all the techniques described in this paper constitute useful tools for the solution of the sparse inverse covariance estimation problem.

## 1 Introduction

*Covariance selection*, first described in [2], has come to refer to the problem of estimating a normal distribution that has a sparse inverse covariance matrix $\mathbf{P}$, whose non-zero entries correspond to edges in an associated Gaussian Markov Random Field, [3]. A popular approach to covariance selection is to maximize an $\ell_1$ penalized log likelihood objective, [4]. This approach has also been applied to related problems, such as sparse multivariate regression with covariance estimation, [5], and covariance selection under a Kronecker product structure, [6]. In this paper, we consider the same objective function as in these papers, and present several Newton-like algorithms for minimizing it.

Following [4, 7, 8], we state the problem as

$$\mathbf{P}^* = \arg\max_{\mathbf{P} \succ 0} \log \det(\mathbf{P}) - \text{trace}(\mathbf{SP}) - \lambda \|\text{vec}(\mathbf{P})\|_1, \tag{1}$$

where $\lambda$ is a (fixed) regularization parameter,

$$\mathbf{S} = \frac{1}{N} \sum_{i=1}^{N} (\mathbf{x}_i - \boldsymbol{\mu})(\mathbf{x}_i - \boldsymbol{\mu})^T \tag{2}$$

is the empirical sample covariance, $\boldsymbol{\mu}$ is known, the $\mathbf{x}_i \in \mathbb{R}^n$ are assumed to be independent, identically distributed samples, and $\text{vec}(\mathbf{P})$ defines a vector in $\mathbb{R}^{n^2}$ obtained by stacking the columns of $\mathbf{P}$. We recast (1) as the minimization problem

$$\min_{P \succ 0} F(\mathbf{P}) \overset{\text{def}}{=} L(\mathbf{P}) + \lambda \|\text{vec}(\mathbf{P})\|_1, \tag{3}$$

where $L$ is the negative log likelihood function

$$L(\mathbf{P}) = -\log \det(\mathbf{P}) + \operatorname{trace}(\mathbf{SP}). \tag{4}$$

The convex problem (3) has a unique solution $\mathbf{P}^*$ that satisfies the optimality conditions [7]

$$\mathbf{S} - [\mathbf{P}^*]^{-1} + \lambda \mathbf{Z}^* = 0, \tag{5}$$

where

$$\mathbf{Z}^*_{ij} = \begin{cases} 1 & \text{if } P^*_{ij} > 0 \\ -1 & \text{if } P^*_{ij} < 0 \\ \alpha \in [-1,1] & \text{if } P^*_{ij} = 0. \end{cases}$$

We note that $\mathbf{Z}^*$ solves the dual problem

$$\mathbf{Z}^* = \arg\max_{\substack{\|\operatorname{vec}(\mathbf{Z})\|_\infty \leq 1 \\ \mathbf{S} + \lambda \mathbf{Z} \succ 0}} U(\mathbf{Z}), \qquad U(\mathbf{Z}) = -\log \det(\mathbf{S} + \lambda \mathbf{Z}) + n. \tag{6}$$

The main contribution of this paper is to propose two classes of second-order methods for solving problem (3). The first class employs a piecewise quadratic model in the step computation, and can be seen as a generalization of the sequential quadratic programming method for nonlinear programming [9]; the second class minimizes a smooth quadratic model of $F$ over a chosen orthant face in $\mathbb{R}^{n^2}$. We argue that both types of methods constitute useful tools for solving the sparse inverse covariance matrix estimation problem.

An overview of recent work on the sparse inverse covariance estimation problem is given in [10, 11]. First-order methods proposed include block-coordinate descent approaches, such as COVSEL, [4, 8] and GLASSO [12], greedy coordinate descent, known as SINCO [13], projected subgradient methods PSM [14], first order optimal gradient ascent [15], and the alternating linearization method ALM [16]. Second-order methods include the inexact interior point method IPM proposed in [17], and the coordinate relaxation method described in [1] and implemented in the QUIC software. It is reported in [1] that QUIC is significantly faster than the ALM, GLASSO, PSM, SINCO and IPM methods. We compare the algorithms presented in this paper to the method implemented in QUIC.

## 2   Newton Methods

We can define a Newton iteration for problem (1) by constructing a quadratic, or piecewise quadratic, model of $F$ using first and second derivative information. It is well known [4] that the derivatives of the log likelihood function (4) are given by

$$\mathbf{g} \overset{\text{def}}{=} L'(\mathbf{P}) = \operatorname{vec}(\mathbf{S} - \mathbf{P}^{-1}) \qquad \text{and} \qquad \mathbf{H} \overset{\text{def}}{=} L''(\mathbf{P}) = (\mathbf{P}^{-1} \otimes \mathbf{P}^{-1}), \tag{7}$$

where $\otimes$ denotes the Kronecker product. There are various ways of using these quantities to define a model of $F$, and each gives rise to a different Newton-like iteration.

In the **Newton-LASSO Method**, we approximate the objective function $F$ at the current iterate $\mathbf{P}_k$ by the piecewise quadratic model

$$q_k(\mathbf{P}) = L(\mathbf{P}_k) + \mathbf{g}_k^\top \operatorname{vec}(\mathbf{P} - \mathbf{P}_k) + \tfrac{1}{2}\operatorname{vec}^\top(\mathbf{P} - \mathbf{P}_k)\mathbf{H}_k \operatorname{vec}(\mathbf{P} - \mathbf{P}_k) + \lambda \|\operatorname{vec}(\mathbf{P})\|_1, \tag{8}$$

where $\mathbf{g}_k = L'(\mathbf{P}_k)$, and similarly for $\mathbf{H}_k$. The trial step of the algorithm is computed as a minimizer of this model, and a backtracking line search ensures that the new iterate lies in the positive definite cone and decreases the objective function $F$. We note that the minimization of $q_k$ is often called the LASSO problem [18] in the case when the unknown is a vector.

It is advantageous to perform the minimization of (8) in a reduced space; see e.g. [11] and the references therein. Specifically, at the beginning of the $k$-th iteration we define the set $\mathcal{F}_k$ of (free) variables that are allowed to move, and the active set $\mathcal{A}_k$. To do so, we compute the steepest descent for the function $F$, which is given by

$$-(\mathbf{g}_k + \lambda \mathbf{Z}_k),$$

where

$$(\mathbf{Z}_k)_{ij} = \begin{cases} 1 & \text{if } (\mathbf{P}_k)_{ij} > 0 \\ -1 & \text{if } (\mathbf{P}_k)_{ij} < 0 \\ -1 & \text{if } (\mathbf{P}_k)_{ij} = 0 \text{ and } [\mathbf{g}_k]_{ij} > \lambda \\ 1 & \text{if } (\mathbf{P}_k)_{ij} = 0 \text{ and } [\mathbf{g}_k]_{ij} < -\lambda \\ -\tfrac{1}{\lambda}[\mathbf{g}_k]_{ij} & \text{if } (\mathbf{P}_k)_{ij} = 0 \text{ and } |[\mathbf{g}_k]_{ij}| \leq \lambda. \end{cases} \tag{9}$$

The sets $\mathcal{F}_k, \mathcal{A}_k$ are obtained by considering a small step along this steepest descent direction, as this guarantees descent in $q_k(\mathbf{P})$. For variables satisfying the last condition in (9), a small perturbation of $\mathbf{P}_{ij}$ will not decrease the model $q_k$. This suggests defining the active and free sets of variables at iteration $k$ as

$$\mathcal{A}_k = \{(i,j)|(\mathbf{P}_k)_{ij} = 0 \text{ and } |[\mathbf{g}_k]_{ij}| \leq \lambda\}, \qquad \mathcal{F}_k = \{(i,j)|(\mathbf{P}_k)_{ij} \neq 0 \text{ or } |[\mathbf{g}_k]_{ij}| > \lambda\}. \quad (10)$$

The algorithm minimizes the model $q_k$ over the set of free variables. Let us define $\mathbf{p}_{\mathcal{F}} = \text{vec}(\mathbf{P})_{\mathcal{F}}$, to be the free variables, and let $\mathbf{p}_{k\mathcal{F}} = \text{vec}_{\mathcal{F}}(\mathbf{P}_k)$ denote their value at the current iterate – and similarly for other quantities. Let us also define $\mathbf{H}_{k\mathcal{F}}$ to be the matrix obtained by removing from $\mathbf{H}_k$ the columns and rows corresponding to the active variables (with indices in $\mathcal{A}_k$). The reduced model is given by

$$q_{\mathcal{F}}(\mathbf{P}) = L(\mathbf{P}_k) + \mathbf{g}_{k\mathcal{F}}^{\top}(\mathbf{p}_{\mathcal{F}} - \mathbf{p}_{k\mathcal{F}}) + \tfrac{1}{2}(\mathbf{p}_{\mathcal{F}} - \mathbf{p}_{k\mathcal{F}})^{\top}\mathbf{H}_{k\mathcal{F}}(\mathbf{p}_{\mathcal{F}} - \mathbf{p}_{k\mathcal{F}}) + \lambda\|\mathbf{p}_{\mathcal{F}}\|_1. \quad (11)$$

The search direction $\mathbf{d}$ is defined by

$$\mathbf{d} = \begin{bmatrix} \mathbf{d}_{\mathcal{F}} \\ \mathbf{d}_{\mathcal{A}} \end{bmatrix} = \begin{bmatrix} \hat{\mathbf{p}}_{\mathcal{F}} - \mathbf{p}_{k\mathcal{F}} \\ 0 \end{bmatrix}, \quad (12)$$

where $\hat{\mathbf{p}}_{\mathcal{F}}$ is the minimizer of (11). The algorithm performs a line search along the direction $\mathbf{D} = \text{mat}(\mathbf{d})$, where the operator $\text{mat}(\mathbf{d})$ satisfies $\text{mat}(\text{vec}(\mathbf{D})) = \mathbf{D}$. The line search starts with the unit steplength and backtracks, if necessary, to obtain a new iterate $\mathbf{P}_{k+1}$ that satisfies the sufficient decrease condition and positive definiteness (checked using a Cholesky factorization):

$$F(\mathbf{P}_{k+1}) - F(\mathbf{P}_k) < \sigma\left(q_{\mathcal{F}}(\mathbf{P}_{k+1}) - q_{\mathcal{F}}(\mathbf{P}_k)\right) \quad \text{and } \mathbf{P}_{k+1} \succ 0, \quad (13)$$

where $\sigma \in (0,1)$.

It is suggested in [1] that coordinate descent is the most effective iteration for solving the LASSO problem (11). We claim, however, that other techniques merit careful investigation. These include gradient projection [19] and iterative shrinkage thresholding algorithms, such as ISTA [20] and FISTA [21]. In section 3 we describe a Newton-LASSO method that employs the FISTA iteration.

Convergence properties of the Newton-LASSO method that rely on the exact solution of the LASSO problem (8) are given in [22]. In practice, it is more efficient to solve problem (8) inexactly, as discussed in section 6. The convergence properties of inexact Newton-LASSO methods will be the subject of a future study.

The **Orthant-Based Newton method** computes steps by solving a *smooth* quadratic approximation of $F$ over an appropriate orthant – or more precisely, over an orthant face in $\mathbb{R}^{n^2}$. The choice of this orthant face is done, as before, by computing the steepest descent direction of $F$, and is characterized by the matrix $\mathbf{Z}_k$ in (9). Specifically the first four conditions in (9) identify an orthant in $\mathbb{R}^{n^2}$ where variables are allowed to move, while the last condition in (9) determines the variables to be held at zero. Therefore, the sets of free and active variables are defined as in (10). If we define $\mathbf{z}_{\mathcal{F}} = \text{vec}_{\mathcal{F}}(\mathbf{Z})$, then the quadratic model of $F$ over the current orthant face is given by

$$Q_{\mathcal{F}}(\mathbf{P}) = L(\mathbf{P}_k) + \mathbf{g}_{\mathcal{F}}^{\top}(\mathbf{p}_{\mathcal{F}} - \mathbf{p}_{k\mathcal{F}}) + \frac{1}{2}(\mathbf{p}_{\mathcal{F}} - \mathbf{p}_{k\mathcal{F}})^{\top}\mathbf{H}_{\mathcal{F}}(\mathbf{p}_{\mathcal{F}} - \mathbf{p}_{k\mathcal{F}}) + \lambda\mathbf{z}_{\mathcal{F}}^{\top}\mathbf{p}_{\mathcal{F}}. \quad (14)$$

The minimizer is $\mathbf{p}_{\mathcal{F}}^* = \mathbf{p}_{k\mathcal{F}} - \mathbf{H}_{\mathcal{F}}^{-1}(\mathbf{g}_{\mathcal{F}} + \lambda\mathbf{z}_{\mathcal{F}})$, and the step of the algorithm is given by

$$\mathbf{d} = \begin{bmatrix} \mathbf{d}_{\mathcal{F}} \\ \mathbf{d}_{\mathcal{A}} \end{bmatrix} = \begin{bmatrix} \mathbf{p}_{\mathcal{F}}^* - \mathbf{p}_{k\mathcal{F}} \\ 0 \end{bmatrix}. \quad (15)$$

If $\mathbf{p}_{k\mathcal{F}} + \mathbf{d}$ lies outside the current orthant, we project it onto this orthant and perform a backtracking line search to obtain the new iterate $\mathbf{P}_{k+1}$, as discussed in section 4.

The orthant-based Newton method therefore moves from one orthant face to another, taking advantage of the fact that $F$ is smooth in every orthant in $\mathbb{R}^{n^2}$. In Figure 1 we compare the two methods discussed so far.

The optimality conditions (5) show that $\mathbf{P}^*$ is diagonal when $\lambda \geq |S_{ij}|$ for all $i \neq j$, and given by $(\text{diag}(\mathbf{S}) + \lambda\mathbf{I})^{-1}$. This suggests that a good choice for the initial value (for any value of $\lambda > 0$) is

$$\mathbf{P}_0 = (\text{diag}(\mathbf{S}) + \lambda\mathbf{I})^{-1}. \quad (16)$$

**Method NL** (Newton-LASSO)

Repeat:

1. *Phase I:* Determine the sets of fixed and free indices $\mathcal{A}_k$ and $\mathcal{F}_k$, using (10).

2. *Phase II:* Compute the Newton step $\mathbf{D}$ given by (12), by minimizing the piecewise quadratic model (11) for the free variables $\mathcal{F}_k$.

3. *Globalization:* Choose $\mathbf{P}_{k+1}$ by performing an Armijo backtracking line search starting from $\mathbf{P}_k + \mathbf{D}$.

4. $k \leftarrow k + 1$.

**Method OBN** (Orthant-Based Newton)

Repeat:

1. *Phase I:* Determine the active orthant face through the matrix $Z_k$ given in (9).

2. *Phase II:* Compute the Newton direction $\mathbf{D}$ given by (15), by minimizing the smooth quadratic model (14) for the free variables $\mathcal{F}_k$.

3. *Globalization:* Choose $\mathbf{P}_{k+1}$ in the current orthant by a *projected* backtracking line search starting from $\mathbf{P}_k + \mathbf{D}$.

4. $k \leftarrow k + 1$.

Figure 1: Two classes of Newton methods for the inverse covariance estimation problem (3).

Numerical experiments indicate that this choice is advantageous for all methods considered.

A popular orthant based method for the case when the unknown is a vector is OWL [23]; see also [11]. Rather than using the Hessian (7), OWL employs a quasi-Newton approximation to minimize the reduced quadratic, and applies an alignment procedure to ensure descent. However, for reasons that are difficult to justify the OWL step employs the reduced *inverse Hessian* (as apposed to the inverse of the reduced Hessian), and this can give steps of poor quality. We have dispensed with the alignment as it is not needed in our experiments. The convergence properties of OBM methods are the subject of a future study (we note in passing that the convergence proof given in [23] is not correct).

## 3 A Newton-LASSO Method with FISTA Iteration

Let us consider a particular instance of the Newton-LASSO method that employs the Fast Iterative Shrinkage Thresholding Algorithm FISTA [21] to solve the reduced subproblem (11). We recall that for the problem

$$\min_{\mathbf{x} \in \mathbb{R}^{n^2}} f(\mathbf{x}) + \lambda \|\mathbf{x}\|_1, \tag{17}$$

where $f$ is a smooth convex quadratic function, the ISTA iteration [20] is given by

$$\mathbf{x}_i = S_{\lambda/c}\left(\hat{\mathbf{x}}_i - \frac{1}{c}\nabla f(\hat{\mathbf{x}}_i)\right), \tag{18}$$

where $c$ is a constant such that $c\mathbf{I} - f''(\mathbf{x}) \succ 0$, and the FISTA acceleration is given by

$$\hat{\mathbf{x}}_{i+1} = \mathbf{x}_i + \frac{t_i - 1}{t_{i+1}}(\mathbf{x}_i - \mathbf{x}_{i-1}), \tag{19}$$

where $\hat{\mathbf{x}}_1 = \mathbf{x}_0$, $t_1 = 1$, $t_{i+1} = \left(1 + \sqrt{1 + 4t_i^2}\right)/2$. Here $S_{\lambda/c}$ denotes the soft thresholding operator given by

$$(S_\sigma(\mathbf{y}))_i = \begin{cases} 0 & \text{if } |y_i| \leq \sigma, \\ y_i - \sigma\,\text{sign}(y_i) & \text{otherwise.} \end{cases}$$

We can apply the ISTA iteration (18) to the reduced quadratic in (11) starting from $\mathbf{x}_0 = \text{vec}_{\mathcal{F}_k}(\mathbf{X}_0)$ (which is not necessarily equal to $\mathbf{p}_k = \text{vec}_{\mathcal{F}_k}(\mathbf{P}_k)$). Substituting in the expressions for the first and second derivative in (7) gives

$$\begin{aligned} \mathbf{x}_i &= S_{\lambda/c}\left(\text{vec}_{\mathcal{F}_k}(\hat{\mathbf{X}}_i) - \frac{1}{c}\left(\mathbf{g}_{k\mathcal{F}_k} + \mathbf{H}_{k\mathcal{F}_k}\text{vec}_{\mathcal{F}_k}(\hat{\mathbf{X}}_i - \mathbf{P}_k)\right)\right) \\ &= S_{\lambda/c}\left(\text{vec}_{\mathcal{F}_k}(\hat{\mathbf{X}}_i) - \frac{1}{c}\text{vec}_{\mathcal{F}_k}(\mathbf{S} - 2\mathbf{P}_k^{-1} + \mathbf{P}_k^{-1}\hat{\mathbf{X}}_i\mathbf{P}_k^{-1})\right), \end{aligned}$$

where the constant $c$ should satisfy $c > 1/(\text{eig}_{\min}\mathbf{P}_k)^2$. The FISTA acceleration step is given by (19). Let $\bar{\mathbf{x}}$ denote the free variables part of the (approximate) solution of (11) obtained by the FISTA iteration. Phase I of the Newton-LASSO-FISTA method selects the free and active sets, $\mathcal{F}_k, \mathcal{A}_k$, as indicated by (10). Phase II, applies the FISTA iteration to the reduced problem (11), and sets $\mathbf{P}_{k+1} \leftarrow \text{mat}\begin{pmatrix} \bar{\mathbf{x}} \\ 0 \end{pmatrix}$. The computational cost of $K$ iterations of the FISTA algorithm is $O(Kn^3)$.

## 4 An Orthant-Based Newton-CG Method

We now consider an orthant-based Newton method in which a quadratic model of $F$ is minimized approximately using the conjugate gradient (CG) method. This approach is attractive since, in addition to the usual advantages of CG (optimal Krylov iteration, flexibility), each CG iteration can be efficiently computed by exploiting the structure of the Hessian matrix in (7).

Phase I of the orthant-based Newton-CG method computes the matrix $\mathbf{Z}_k$ given in (9), which is used to identify an orthant face in $\mathbb{R}^{n^2}$. Variables satisfying the last condition in (9) are held at zero and their indices are assigned to the set $\mathcal{A}_k$, while the rest of the variables are assigned to $\mathcal{F}_k$ and are allowed to move according to the signs of $\mathbf{Z}_k$: variables with $(\mathbf{Z}_k)_{ij} = 1$ must remain non-negative, and variables with $(\mathbf{Z}_k)_{ij} = -1$ must remain non-positive.

Having identified the current orthant face, phase II of the method constructs the quadratic model $Q_{\mathcal{F}}$ in the free variables, and computes an approximate solution by means of the conjugate gradient method, as described in Algorithm 1.

---

**Conjugate Gradient Method for Problem** (14)
**input** : Gradient $\mathbf{g}$, orthant indicator $\mathbf{z}$, current iterate $\mathbf{P}_0$, maximum steps $K$, residual tolerance $\epsilon_r$, and the regularization parameter $\lambda$.
**output**: Approximate Newton direction $\mathbf{d} = \text{cg}(\mathbf{P}_0, \mathbf{g}, \mathbf{z}, \lambda, K)$

$n = \text{size}(\mathbf{P}_0, 1)$, $\mathbf{G} = \text{mat}(\mathbf{g})$, $\mathbf{Z} = \text{mat}(\mathbf{z})$;
$\mathcal{A} = \{(i, j) : [\mathbf{P}_0]_{ij} = 0 \ \& \ |\mathbf{G}_{ij}| \le \lambda\}$;
$\mathbf{B} = \mathbf{P}_0^{-1}$, $\mathbf{X}_0 = \mathbf{0}_{n \times n}$, $\mathbf{x}_0 = \text{vec}(\mathbf{X}_0)$;
$\mathbf{R}_0 = -(\mathbf{G} + \lambda\mathbf{Z})$, $[\mathbf{R}_0]_{\mathcal{A}} \leftarrow 0$;
$k = 0$, $\mathbf{q}_0 = \mathbf{r}_0 = \text{vec}(\mathbf{R}_0)$; $\quad (\therefore [\mathbf{r}_0]_{\mathcal{F}} = \mathbf{v}_{\mathcal{F}})$
**while** $k \le \min(n^2, K)$ *and* $\|\mathbf{r}_k\| > \epsilon_r$ **do**
$\quad$ $\mathbf{Q}_k = \text{reshape}(\mathbf{q}_k, n, n)$;
$\quad$ $\mathbf{Y}_k = \mathbf{B}\mathbf{Q}_k\mathbf{B}$, $[\mathbf{Y}_k]_{\mathcal{A}} \leftarrow 0$, $\mathbf{y}_k = \text{vec}(\mathbf{Y}_k)$;
$\quad$ $\alpha_k = \frac{\mathbf{r}_k^\top \mathbf{r}_k}{\mathbf{q}_k^\top \mathbf{y}_k}$;
$\quad$ $\mathbf{x}_{k+1} = \mathbf{x}_k + \alpha_k\mathbf{q}_k$;
$\quad$ $\mathbf{r}_{k+1} = \mathbf{r}_k - \alpha_k\mathbf{y}_k$;
$\quad$ $\beta_k = \frac{\mathbf{r}_{k+1}^\top \mathbf{r}_{k+1}}{\mathbf{r}_k^\top \mathbf{r}_k}$;
$\quad$ $\mathbf{q}_{k+1} = \mathbf{r}_{k+1} + \beta_k\mathbf{q}_k$;
$\quad$ $k \leftarrow k + 1$;
**end**
**return** $\mathbf{d} = \mathbf{x}_{k+1}$

**Algorithm 1:** CG Method for Minimizing the Reduced Model $Q_{\mathcal{F}}$.

---

The search direction of the method is given by $\mathbf{D} = \text{mat}(\mathbf{d})$, where $\mathbf{d}$ denotes the output of Algorithm 1. If the trial step $\mathbf{P}_k + \mathbf{D}$ lies in the current orthant, it is the optimal solution of (14). Otherwise, there is at least one index such that

$$(i, j) \in \mathcal{A}_k \text{ and } [L'(\mathbf{P}_k + \mathbf{D})]_{ij} \notin [-\lambda, \lambda], \quad \text{or} \quad (i, j) \in \mathcal{F}_k \text{ and } (\mathbf{P}_k + \mathbf{D})_{ij}\mathbf{Z}_{ij} < 0.$$

In this case, we perform a projected line search to find a point in the current orthant that yields a decrease in $F$. Let $\Pi(\cdot)$ denote the orthogonal projection onto the orthant face defined by $\mathbf{Z}_k$, i.e.,

$$\Pi(\mathbf{P}_{ij}) = \begin{cases} \mathbf{P}_{ij} & \text{if } \text{sign}(\mathbf{P}_{ij}) = \text{sign}(\mathbf{Z}_k)_{ij} \\ 0 & \text{otherwise}. \end{cases} \tag{20}$$

The line search computes a steplength $\alpha_k$ to be the largest member of the sequence $\{1, 1/2, \ldots, 1/2^i, \ldots\}$ such that

$$F(\Pi(\mathbf{P}_k + \alpha_k \mathbf{D})) \leq F(\mathbf{P}_k) + \sigma \widetilde{\nabla} F(\mathbf{P}_k)^T \left(\Pi(\mathbf{P}_k + \alpha_k \mathbf{D}) - \mathbf{P}_k\right), \qquad (21)$$

where $\sigma \in (0,1)$ is a given constant and $\widetilde{\nabla} F$ denotes the minimum norm subgradient of $F$. The new iterate is defined as $\mathbf{P}_{k+1} = \Pi(\mathbf{P}_k + \alpha_k \mathbf{D})$.

The conjugate gradient method requires computing matrix-vector products involving the reduced Hessian, $\mathbf{H}_{k\mathcal{F}}$. For our problem, we have

$$
\begin{aligned}
\mathbf{H}_{k\mathcal{F}}(\mathbf{p}_{\mathcal{F}} - \mathbf{p}_{k\mathcal{F}}) &= \left[\mathbf{H}_k \left(\begin{smallmatrix}\mathbf{P}_{\mathcal{F}} - \mathbf{P}_{k\mathcal{F}} \\ \mathbf{0}\end{smallmatrix}\right)\right]_{\mathcal{F}} \\
&= \left[\mathbf{P}_k^{-1} \mathrm{mat}\left(\begin{smallmatrix}\mathbf{P}_{\mathcal{F}} - \mathbf{P}_{k\mathcal{F}} \\ \mathbf{0}\end{smallmatrix}\right) \mathbf{P}_k^{-1}\right]_{\mathcal{F}}.
\end{aligned}
\qquad (22)
$$

The second line follows from the identity $(\mathbf{A} \otimes \mathbf{B})\mathrm{vec}(\mathbf{C}) = \mathrm{vec}(\mathbf{BCA}^\top)$. The cost of performing $K$ steps of the CG algorithm is $O(Kn^3)$ operations, and $K = n^2$ steps is needed to guarantee an exact solution. Our practical implementation computes a small number of CG steps relative to n, $K = O(1)$, and as a result the search direction is not an accurate approximation of the true Newton step. However, such inexact Newton steps achieve a good balance between the computational cost and the quality of the direction.

## 5  Quasi-Newton Methods

The methods considered so far employ the exact Hessian of the likelihood function $L$, but one can also approximate it using (limited memory) quasi-Newton updating. At first glance it may not seem promising to approximate a complicated Hessian like (7) in this manner, but we will see that quasi-Newton updating is indeed effective, provided that we store matrices using the compact limited memory representations [9].

Let us consider an orthant-based method that minimizes the quadratic model (14), where $\mathbf{H}_{\mathcal{F}}$ is replaced by a limited memory BFGS matrix, which we denote by $\mathbf{B}_{\mathcal{F}}$. This matrix is not formed explicitly, but is defined in terms of the difference pairs

$$\mathbf{y}_k = \mathbf{g}_{k+1} - \mathbf{g}_k, \quad \mathbf{s}_k = \mathrm{vec}(\mathbf{P}_{k+1} - \mathbf{P}_k). \qquad (23)$$

It is shown in [24, eq(5.11)] that the minimizer of the model $Q_{\mathcal{F}}$ is given by

$$
\begin{aligned}
\mathbf{p}_{\mathcal{F}}^* &= \mathbf{p}_{\mathcal{F}} + \mathbf{B}_{\mathcal{F}}^{-1}(\lambda \mathbf{z}_{\mathcal{F}} - \mathbf{g}_{\mathcal{F}}) \\
&= \tfrac{1}{\theta}(\lambda \mathbf{z}_{\mathcal{F}} - \mathbf{g}_{\mathcal{F}}) + \tfrac{1}{\theta^2} \mathbf{R}_{\mathcal{F}}^T \mathbf{W} (\mathbf{I} - \tfrac{1}{\theta} \mathbf{M} \mathbf{W}^T \mathbf{R}_{\mathcal{F}} \mathbf{R}_{\mathcal{F}}^T \mathbf{W})^{-1} \mathbf{M} \mathbf{W}^T \mathbf{R}_{\mathcal{F}} (\lambda \mathbf{z}_{\mathcal{F}} - \mathbf{g}_{\mathcal{F}}).
\end{aligned}
\qquad (24)
$$

Here $\mathbf{R}_{\mathcal{F}}$ is a matrix consisting of the set of unit vectors that span the subspace of free variables, $\theta$ is a scalar, $\mathbf{W}$ is an $n^2 \times 2t$ matrix containing the $t$ most recent correction pairs (23), and $\mathbf{M}$ is a $2t \times 2t$ matrix formed by inner products between the correction pairs. The total cost of computing the minimizer $\mathbf{p}_{\mathcal{F}}^*$ is $2t^2|\mathcal{F}| + 4t|\mathcal{F}|$ operations, where $|\mathcal{F}|$ is the cardinality of $\mathcal{F}$. Since the memory parameter $t$ in the quasi-Newton updating scheme is chosen to be a small number, say between 5 and 20, the cost of computing the subspace minimizer (24) is quite affordable. A similar approach was taken in [25] for the related constrained inverse covariance sparsity problem.

We have noted above that OWL, which is an orthant based quasi-Newton method does not correctly approximate the minimizer (24). We note also that quasi-Newton updating can be employed in Newton-LASSO methods, but we do not discuss this approach here for the sake of brevity.

## 6  Numerical Experiments

We generated test problems by first creating a random sparse inverse covariance matrix[1], $\mathbf{\Sigma}^{-1}$, and then sampling data to compute a corresponding non-sparse empirical covariance matrix $\mathbf{S}$. The dimensions, sparsity, and conditioning of the test problems are given along with the results in Table 2. For each data set, we solved problem (3) with $\lambda$ values in the range $[0.01, 0.5]$. The number of samples used to compute the sample covariance matrix was $10n$.

The algorithms we tested are listed in Table 1. With the exception of C:QUIC, all of these algorithms were implemented in MATLAB. Here NL and OBN are abbreviations for the methods in Figure 1. NL-Coord is a MATLAB implementation of the QUIC algorithm that follows the C-version [1]

| Algorithm | Description |
|-----------|-------------|
| NL-FISTA | Newton-LASSO-FISTA method |
| NL-Coord | Newton-LASSO method using coordinate descent |
| OBN-CG-K | Orthant-based Newton-CG method with a limit of K CG iterations |
| OBN-CG-D | OBN-CG-K with K=5 initially and increased by 1 every 3 iterations. |
| OBN-LBFGS | Orthant-based quasi-Newton method (see section 5) |
| ALM* | Alternating linearization method [26]. |
| C:QUIC | The C implementation of QUIC given in [1]. |

Table 1: Algorithms tested. *For ALM, the termination criteria was changed to the $\ell_\infty$ norm and the value of ABSTOL was set to $10^{-6}$ to match the stopping criteria of the other algorithms.

faithfully. We have also used the original C-implementation of QUIC and refer to it as C:QUIC. For the Alternating Linearization Method (ALM) we utilized the MATLAB software available at [26], which implements the first-order method described in [16]. The NL-FISTA algorithm terminated the FISTA iteration when the minimum norm subgradient of the LASSO subproblem $q_{\mathcal{F}}$ became less than $1/10$ of the minimum norm subgradient of $F$ at the previous step.

Let us compare the computational cost of the inner iteration techniques used in the Newton-like methods discussed in this paper. (i) Applying $K$ steps of the FISTA iteration requires $O(Kn^3)$ operations. (ii) Coordinate descent, as implemented in [1], requires $O(Kn|\mathcal{F}|)$ operations for $K$ coordinate descent sweeps through the set of free variables; (iii) Applying $K_{CG}$ iterations of the CG methods costs $O(K_{CG}n^3)$ operations.

The algorithms were terminated when either $10n$ iterations were executed or the minimum norm subgradient of $F$ was sufficiently small, i.e. when $\|\tilde{\nabla}F(\mathbf{P})\|_\infty \leq 10^{-6}$. The time limit of each run was set to 5000 seconds.

The results presented in Table 2 show that the ALM method was never the fastest algorithm, but nonetheless outperformed some second-order methods when the solution was less sparse. The numbers in bold indicate the fastest MATLAB implementation for each problem. As for the other methods, no algorithm appears to be consistently superior to the others, and the best choice may depend on problem characteristics. The Newton-LASSO method with coordinate descent (NL-Coord) is the most efficient when the sparsity level is below 1%, but the methods introduced in this paper, NL-FISTA, OBN-CG and OBN-LBFGS, seem more robust and efficient for problems that are less sparse. Based on these results, OBN-LBFGS appears to be the best choice as a universal solver for the covariance selection problem. The C implementation of the QUIC algorithm is roughly five times faster than its Matlab counterpart (OBN-Coord). C:QUIC was best in the two sparsest conditions, but not in the two densest conditions. We expect that optimized C implementations of the presented algorithms will also be significantly faster. Note also that the crude strategy for dynamically increasing the number of CG-steps in OBN-CG-D was effective, and we expect it could be further improved. Our focus in this paper has been on exploring optimization methods and ideas rather than implementation efficiency. However, we believe the observed trends will hold even for highly optimized versions of all tested algorithms.

| problem | algorithm | λ | 0.5 | | 0.1 | | 0.05 | | 0.01 | |
|---|---|---|---|---|---|---|---|---|---|---|
| | | | iter | time | iter | time | iter | time | iter | time |
| $n = 500$ $\mathrm{Card}(\mathbf{\Sigma}^{-1})$ $= 2.4\%$ | card($\mathbf{P}^*$) | | 0.74% | | 7.27% | | 11.83% | | 32.48% | |
| | cond($\mathbf{P}^*$) | | 8.24 | | 27.38 | | 51.01 | | 118.56 | |
| | NL-FISTA | | 8 | 5.71 | 10 | 22.01 | 11 | **37.04** | 12 | 106.27 |
| | NL-Coord | | 21 | **3.86** | 49 | 100.63 | 66 | 279.69 | 103 | 1885.89 |
| | OBN-CG-5 | | 15 | 4.07 | 97 | 26.24 | 257 | 70.91 | 1221 | 373.63 |
| | OBN-CG-D | | 12 | 3.88 | 34 | **15.41** | 65 | 43.29 | 189 | 275.29 |
| | OBN-LBFGS | | 47 | 5.37 | 178 | 21.92 | 293 | 38.23 | 519 | **84.13** |
| | ALM | | 445 | 162.96 | 387 | 152.76 | 284 | 115.11 | 574 | 219.80 |
| | C:QUIC | | 16 | 0.74 | 41 | 15.62 | 58 | 35.64 | 100 | 206.42 |
| $n = 500$ $\mathrm{Card}(\mathbf{\Sigma}^{-1})$ $= 20.1\%$ | card($\mathbf{P}^*$) | | 0.21% | | 14.86% | | 25.66% | | 47.33% | |
| | cond($\mathbf{P}^*$) | | 3.39 | | 16.11 | | 32.27 | | 99.49 | |
| | NL-FISTA | | 4 | 1.25 | 19 | 13.12 | 15 | 34.53 | 13 | 100.90 |
| | NL-Coord | | 4 | **0.42** | 14 | 19.69 | 21 | 71.51 | 55 | 791.84 |
| | OBN-CG-5 | | 3 | 0.83 | 27 | 7.36 | 101 | 28.40 | 795 | 240.90 |
| | OBN-CG-D | | 3 | 0.84 | 15 | **5.22** | 31 | **14.14** | 176 | 243.55 |
| | OBN-LBFGS | | 9 | 1.00 | 82 | 11.42 | 155 | 23.04 | 455 | **78.33** |
| | ALM | | 93 | 35.75 | 78 | 32.98 | 149 | 61.35 | 720 | 292.43 |
| | C:QUIC | | 6 | 0.19 | 13 | 3.79 | 21 | 11.91 | 56 | 103.58 |
| $n = 1000$ $\mathrm{Card}(\mathbf{\Sigma}^{-1})$ $= 3.5\%$ | card($\mathbf{P}^*$) | | 0.18% | | 6.65% | | 13.19% | | 25.03% | |
| | cond($\mathbf{P}^*$) | | 6.22 | | 18.23 | | 39.59 | | 132.13 | |
| | NL-FISTA | | 7 | 28.20 | 9 | 106.79 | 12 | 203.07 | 12 | 801.79 |
| | NL-Coord | | 9 | **5.23** | 24 | 225.59 | 36 | 951.23 | - | >5000 |
| | OBN-CG-5 | | 9 | 15.34 | 51 | 87.73 | 108 | 198.17 | 1103 | 2026.26 |
| | OBN-CG-D | | 8 | 15.47 | 21 | **51.99** | 39 | 132.38 | 171 | 1584.14 |
| | OBN-LBFGS | | 34 | 18.27 | 111 | 80.02 | 178 | **111.49** | 548 | **384.30** |
| | ALM | | 247 | 617.63 | 252 | 639.49 | 186 | 462.34 | 734 | 1826.29 |
| | C:QUIC | | 10 | 2.38 | 22 | 46.14 | 34 | 186.87 | 72 | 1445.17 |
| $n = 1000$ $\mathrm{Card}(\mathbf{\Sigma}^{-1})$ $= 11\%$ | card($\mathbf{P}^*$) | | 0.10% | | 8.18% | | 18.38% | | 36.34% | |
| | cond($\mathbf{P}^*$) | | 4.20 | | 11.75 | | 26.75 | | 106.34 | |
| | NL-FISTA | | 4 | 9.03 | 7 | 72.21 | 10 | 156.46 | 22 | 554.08 |
| | NL-Coord | | 4 | **2.23** | 12 | 79.71 | 19 | 408.62 | 49 | 4837.46 |
| | OBN-CG-5 | | 3 | 4.70 | 20 | 35.85 | 47 | 83.42 | 681 | 1778.88 |
| | OBN-CG-D | | 3 | 4.61 | 12 | **26.87** | 27 | **78.98** | 148 | 2055.44 |
| | OBN-LBFGS | | 8 | 4.29 | 67 | 40.31 | 124 | 82.51 | 397 | **297.90** |
| | ALM | | 113 | 283.99 | 99 | 255.79 | 106 | 267.02 | 577 | 1448.83 |
| | C:QUIC | | 6 | 1.18 | 11 | 17.42 | 19 | 90.62 | 52 | 1100.72 |
| $n = 2000$ $\mathrm{Card}(\mathbf{\Sigma}^{-1})$ $= 1\%$ | card($\mathbf{P}^*$) | | 0.13% | | 1.75% | | 4.33% | | 14.68% | |
| | cond($\mathbf{P}^*$) | | 7.41 | | 23.71 | | 46.54 | | 134.54 | |
| | NL-FISTA | | 8 | 264.94 | 10 | 1039.08 | 10 | 1490.37 | - | >5000 |
| | NL-Coord | | 14 | **54.33** | 34 | 1178.07 | - | >5000 | - | >5000 |
| | OBN-CG-5 | | 13 | 187.41 | 78 | 896.24 | 203 | 2394.95 | - | >5000 |
| | OBN-CG-D | | 9 | 127.11 | 27 | 532.15 | 43 | 1038.26 | - | >5000 |
| | OBN-LBFGS | | 41 | 115.13 | 155 | **497.31** | 254 | **785.36** | 610 | **2163.12** |
| | ALM | | - | >5000 | - | >5000 | - | >5000 | - | >5000 |
| | C:QUIC | | 11 | 18.07 | 17 | 183.53 | 40 | 818.54 | - | >5000 |
| $n = 2000$ $\mathrm{Card}(\mathbf{\Sigma}^{-1})$ $= 18.7\%$ | card($\mathbf{P}^*$) | | 0.05% | | 1.49% | | 10.51% | | 31.68% | |
| | cond($\mathbf{P}^*$) | | 2.32 | | 4.72 | | 17.02 | | 79.61 | |
| | NL-FISTA | | $\mathbf{P}^* = \mathbf{P}_0$ | | 7 | 153.18 | 9 | 694.93 | 12 | 2852.86 |
| | NL-Coord | | $\mathbf{P}^* = \mathbf{P}_0$ | | 7 | 71.55 | 13 | 1152.86 | - | >5000 |
| | OBN-CG-5 | | $\mathbf{P}^* = \mathbf{P}_0$ | | 6 | **71.54** | 21 | 250.11 | 397 | 4766.69 |
| | OBN-CG-D | | $\mathbf{P}^* = \mathbf{P}_0$ | | 6 | 75.82 | 13 | **188.93** | 110 | 5007.83 |
| | OBN-LBFGS | | $\mathbf{P}^* = \mathbf{P}_0$ | | 26 | 78.34 | 71 | 232.23 | 318 | **1125.67** |
| | ALM | | 52 | 874.22 | 76 | 1262.83 | 106 | 1800.67 | - | >5000 |
| | C:QUIC | | 8 | 10.35 | 8 | 24.65 | 13 | 256.90 | 33 | 3899.68 |

Table 2: Results for 5 Newton-like methods and the QUIC, ALM method.

## Footnotes

[1]http://www.cmap.polytechnique.fr/~aspremon/CovSelCode.html, [7]

# References

[1] C. J. Hsieh, M. A. Sustik, P. Ravikumar, and I. S. Dhillon. Sparse inverse covariance matrix estimation using quadratic approximation. *Advances in Neural Information Processing Systems (NIPS)*, 24, 2011.

[2] A. P. Dempster. Covariance selection. *Biometrics*, 28:157–75, 1972.

[3] J. D. Picka. Gaussian Markov random fields: theory and applications. *Technometrics*, 48(1):146–147, 2006.

[4] O. Banerjee, L. El Ghaoui, A. d'Aspremont, and G. Natsoulis. Convex optimization techniques for fitting sparse Gaussian graphical models. In *ICML*, pages 89–96. ACM, 2006.

[5] A.J. Rothman, E. Levina, and J. Zhu. Sparse multivariate regression with covariance estimation. *Journal of Computational and Graphical Statistics*, 19(4):947–962, 2010.

[6] T. Tsiligkaridis and A. O. Hero III. Sparse covariance estimation under Kronecker product structure. In *ICASSP 2006 Proceedings*, pages 3633–3636, Kyoto, Japan, 2012.

[7] O. Banerjee, L. El Ghaoui, and A. d'Aspremont. Model selection through sparse maximum likelihood estimation for multivariate gaussian or binary data. *The Journal of Machine Learning Research*, 9:485–516, 2008.

[8] A. d'Aspremont, O. Banerjee, and L. El Ghaoui. First-order methods for sparse covariance selection. *SIAM Journal on Matrix Analysis and Applications*, 30(1):56–66, 2008.

[9] J. Nocedal and S. J. Wright. *Numerical Optimization*. Springer Series in Operations Research. 1999.

[10] I. Rish and G. Grabarnik. ELEN E6898 Sparse Signal Modeling (Spring 2011): Lecture 7, Beyond LASSO: Other Losses (Likelihoods). https://sites.google.com/site/eecs6898sparse2011/, 2011.

[11] S. Sra, S. Nowozin, and S. J. Wright. *Optimization for Machine Learning*. MIT Press, 2011.

[12] J. Friedman, T. Hastie, and R. Tibshirani. Sparse inverse covariance estimation with the graphical LASSO. *Biostatistics*, 9(3):432, 2008.

[13] K. Scheinberg and I. Rish. SINCO-a greedy coordinate ascent method for sparse inverse covariance selection problem. Technical report, IBM RC24837, 2009.

[14] J. Duchi, S. Gould, and D. Koller. Projected subgradient methods for learning sparse Gaussians. In *Proc. of the Conf. on Uncertainty in AI*. Citeseer, 2008.

[15] Z. Lu. Smooth optimization approach for sparse covariance selection. *Arxiv preprint arXiv:0904.0687*, 2009.

[16] K. Scheinberg, S. Ma, and D. Goldfarb. Sparse inverse covariance selection via alternating linearization methods. *Arxiv preprint arXiv:1011.0097*, 2010.

[17] L. Li and K. C. Toh. An inexact interior point method for L1-regularized sparse covariance selection. *Mathematical Programming Computation*, 2(3):291–315, 2010.

[18] R. Tibshirani. Regression shrinkage and selection via the LASSO. *Journal of the Royal Statistical Society B*, 58(1):267–288, 1996.

[19] B. T. Polyak. The conjugate gradient method in extremal problems. *U.S.S.R. Computational Mathematics and Mathematical Physics*, 9:94–112, 1969.

[20] I. Daubechies, M. Defrise, and C. De Mol. An iterative thresholding algorithm for linear inverse problems with a sparsity constraint. *Communications on pure and applied mathematics*, 57(11):1413–1457, 2004.

[21] A. Beck and M. Teboulle. A fast iterative shrinkage-thresholding algorithm for linear inverse problems. *SIAM Journal on Imaging Sciences*, 2(1):183–202, 2009.

[22] P. Tseng and S. Yun. A coordinate gradient descent method for nonsmooth separable minimization. *Mathematical Programming*, 117(1):387–423, 2009.

[23] G. Andrew and J. Gao. Scalable training of L1-regularized log-linear models. In *ICML*, pages 33–40. ACM, 2007.

[24] R. H. Byrd, P. Lu, J. Nocedal, and C. Zhu. A limited memory algorithm for bound constrained optimization. *SIAM Journal on Scientific Computing*, 16(5):1190–1208, 1995.

[25] J. Dahl, V. Roychowdhury, and L. Vandenberghe. Maximum likelihood estimation of gaussian graphical models: numerical implementation and topology selection. *UCLA Preprint*, 2005.

[26] S. Boyd, N. Parikh, E. Chu, B. Peleato, and J. Eckstein. Matlab scripts for alternating direction method of multipliers. Technical report, http://www.stanford.edu/ boyd/papers/admm/, 2012.

